# Real-time autonomous robot navigation using VLSI neural networks

Lionel Tarassenko    Michael Brownlow    Gillian Marshall*        Jon Tombs
Department of Engineering Science
Oxford University, Oxford, OX1 3PJ, UK

**Alan Murray**

Department of Electrical Engineering
Edinburgh University, Edinburgh, EH9 3JL, UK

## Abstract

We describe a real time robot navigation system based on three VLSI neural network modules. These are a resistive grid for path planning, a nearest-neighbour classifier for localization using range data from a time-of-flight infra-red sensor and a sensory-motor associative network for dynamic obstacle avoidance.

## 1  INTRODUCTION

There have been very few demonstrations of the application of VLSI neural networks to *real world* problems. Yet there are many signal processing, pattern recognition or optimization problems where a large number of competing hypotheses need to be explored in parallel, most often in real time. The massive parallelism of VLSI neural network devices, with one multiplier circuit per synapse, is ideally suited to such problems. In this paper, we present preliminary results from our design for a real time robot navigation system based on VLSI neural network modules. This is a

real world problem which has not been *fully* solved by traditional AI methods; even when partial solutions have been proposed and implemented, these have required vast computational resources, usually remote from the robot and linked to it via an umbilical cord.

## 2   OVERVIEW

The aim of our work is to develop an autonomous vehicle capable of *real–time* navigation, including obstacle avoidance, in a known indoor environment. The obstacles may be permanent (static) or unexpected and dynamic (for example, in an automated factory environment, the walls and machines are permanent but people, other moving vehicles and packages are not.) There are three neural network modules at the heart of our navigation system: a localization module (to determine, at any time, the robot's position within the environment), an obstacle detection module and a path planning module (to compute a path to the goal which avoids obstacles). These modules perform *low–level processing in real time* which can then be decoupled from higher level processing to be carried out by a simple controller. It is our view that such a hybrid system is the best way to realise the computational potential of artificial neural networks for solving a real world problem such as this without compromising overall system performance.

A short description of each module is now given. In each case, the general principles are first outlined and, where applicable, the results of our preliminary work are then reported.

## 3   PATH PLANNING

The use of resistive grids for parallel analog computation was first suggested by Horn in the mid-seventies (Horn, 1974) and the idea has since been exploited by Mead and co-workers, for example in a silicon retina (Mead and Mahowald, 1988). Although these resistive grids cannot be said to be neural networks in the conventional sense, they also perform parallel analog computation and they have the same advantages, in terms of speed and fault-tolerance, as any hardware realisation of neural networks.

We have taken the resistive grid concept and applied it to the path planning problem, here taken to be the computation of an obstacle-avoiding path, in a structured environment, from the robot's initial (or present) position (P) to its goal (G). In our approach, the robot's working domain is discretized and mapped onto a resistive grid of hexagonal or rectangular cells – see Figure 1 which shows the test environment for Autonomous Guided Vehicles (AGV's) in the Oxford Robotics Laboratory. Each resistor in the grid has a value of $R_0$, unless it is part of a region of the grid corresponding to an obstacle, in which case its resistance is infinite ($R_\infty$).

The principle of the method is perhaps best understood by considering a continuous analog of the resistive grid (for example, a sheet of material of uniform resistivity in which holes have been cut to represent the obstacles). The current streamlines resulting from the application of an external source between P and G skirt around the obstacles; if we follow one of these streamlines from P to G, we will obtain a *guaranteed* collision-free path since current cannot flow into the obstacles (Tarassenko and

Blake, 1991). For simple cases such as circularly symmetric conductivity distributions in 2D, Laplace's equation can be solved in order to calculate the value of the potential $V$ at every point within the workspace. Following a current streamline is then simply a matter of performing gradient descent in $V$.

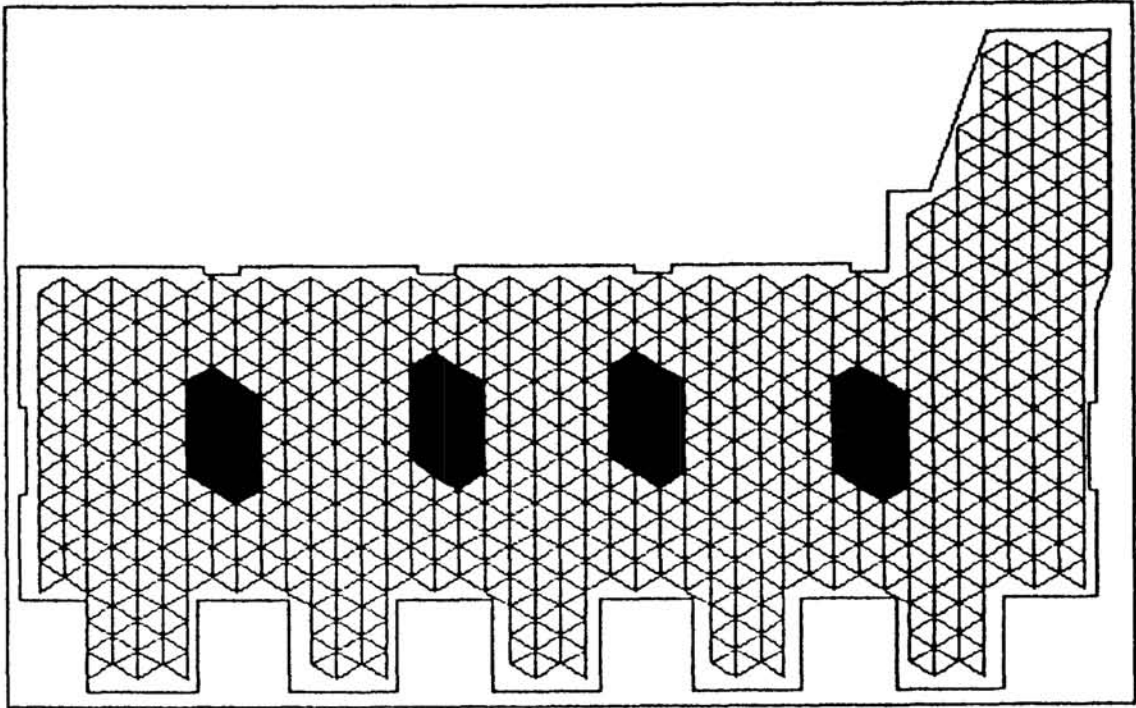

Figure 1: The Oxford test environment for AGV's mapped out as a hexagonal resistive grid. The resistors corresponding to the four pillars in the middle are open circuits. Note that the pillars are enlarged in their grid representation in order to take into account the mobile robot's finite size.

It is not possible, however, to solve Laplace's equation analytically for realistic environments. With the resistive grid, the problem is discretized and mapped onto a hardware representation which can be implemented in VLSI. As soon as an external source of power is connected between P and G, the resistive network settles into the state of least power dissipation and the node voltages can be read out (hardware computation of Kirchhoff's equations). The path from P to G is computed incrementally from *local* voltage measurements: for each node, the next move is identified by measuring the voltage drop $\Delta V_n$ between that node and each of its nearest neighbours ($n = 6$ for a hexagonal grid) and then selecting the node corresponding to $(\Delta V_n)_{max}$. This is illustrated by the example of a robot in a maze (Figure 2). As above, the resistors shown shaded are open circuits whilst all other resistors are set to be equal to $R_0$. The robot is initially placed at the centre of the maze (P) and a path has to be found to the goal in the top left-hand corner (G). The solid line shows the path resulting from a *single application* of the voltage between P and G. The dotted line shows the (optimal) path computed by re-applying the

voltage at every node as the robot moves towards the goal. As already indicated, this is actually how we intend to use the resistive grid planner in practice, since this approach also allows us to re-compute the robot's path whenever unexpected obstacles appear in the environment (see Section 5).

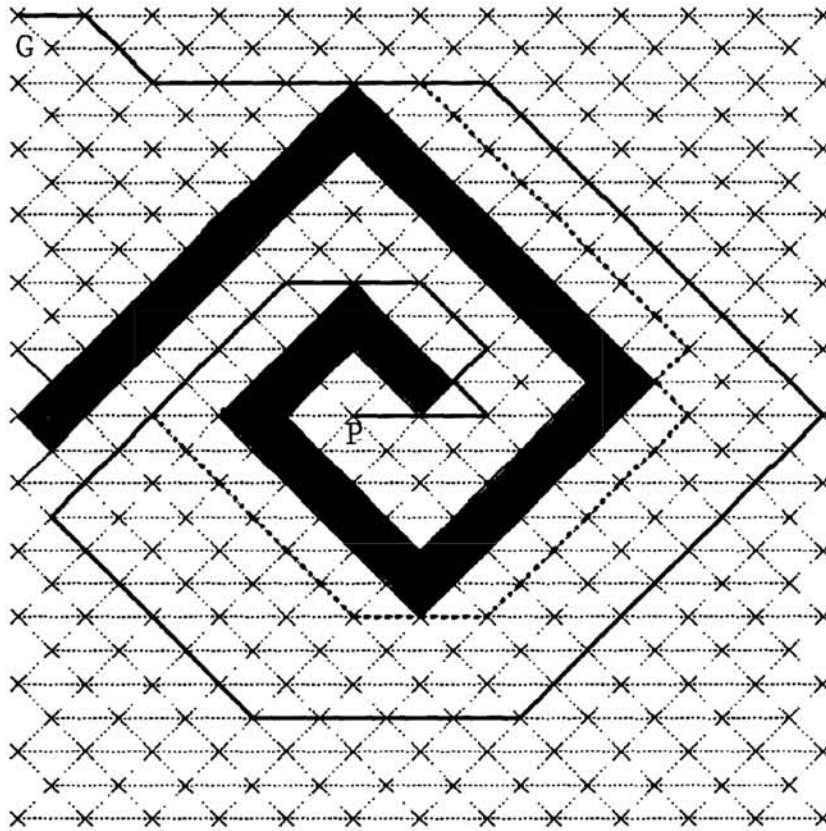

Figure 2: Path from middle of maze (P) to top left-hand corner (G)

## 3.1  VLSI IMPLEMENTATION

The VLSI implementation of the resistive grid method will allow us to solve the path planning for complex environments in real time. MOS switches are ideal implementations of the binary resistors in the grid. Each transistor can be programmed to be either open ($R_\infty$) or closed ($R_0$) from a RAM cell connected to its gate. With the incremental computation of the path described above, the selection of the next move is a matter of identifying the largest of six voltages. Of course, the nearest neighbour voltages and that of the P node could be read out through an A/D converter and the decision made off-chip. We favour a full hardware solution instead, whereby the maximum voltage difference is directly identified on-chip.

## 4  LOCALIZATION

The autonomous robot should at any time be able to work out its position in the workspace so that the path to the goal can be updated if required. The grid representation of the environment used for the path planner can also be employed

for localization purposes, in which case localization becomes, in the first instance, a matter of identifying the nearest node in the grid at any time during navigation.

This task can be performed by harnessing the pattern recognition capabilities of neural networks. The room environment is learnt by recording a 360° range scan at every node during a training phase prior to navigation. During navigation, the nearest node is identified using a minimum-distance classifier implemented on a single-layer neural network working on dense input data (one range value every 3°, say). In order to solve the localization problem in real-time, we have designed a time-of-flight *optical* rangefinder, which uses near infra-red light, amplitude-modulated at a frequency of just above 5 MHz, together with a heterodyne mixing technique. Our design is capable of resolving phase shifts in the received light signal of the order of 0.1° over a 50 dB dynamic range.

The rotating optical scanner gives a complete 360° scan approximately every second during navigation. The minimum-distance classifier is used to compare this scan $\mathbf{x}$ with the $k$ patterns $\mathbf{u}_i$ recorded at each node during training. If we use a Euclidean metric for the comparison, this is equivalent to identifying the pattern $\mathbf{u}_i$ for which:

$$\| \mathbf{x} - \mathbf{u}_i \|^2 = \| \mathbf{x} \|^2 - 2\mathbf{u}_i^T \mathbf{x} + \| \mathbf{u}_i \|^2 \tag{1}$$

is a minimum. The first term in the above equation is the same for all $i$ and can be ignored. We can therefore write:

$$g_i(\mathbf{x}) = -\frac{1}{2}(-2\mathbf{w}_i^T \mathbf{x} + \mathbf{u}_i^2) = \mathbf{w}_i^T \mathbf{x} + w_{i0} \tag{2}$$

where $g_i(\mathbf{x})$ is a linear discriminant function, $\mathbf{w}_i = \mathbf{u}_i$ and $w_{i0} = -\frac{1}{2}\mathbf{u}_i^2$. Thus each $\mathbf{w}_i$ vector is one of the learnt patterns $\mathbf{u}_i$ and the discriminant $g_i(\mathbf{x})$ matches the input $\mathbf{x}$ with $\mathbf{u}_i$, point by point. If we let $\mathbf{w}_i = \{T_{ij}\}$ and $\mathbf{x} = \{V_j\}$ and assume that there are $n$ range values in each scan, then we can write:

$$g_i(\mathbf{x}) = \sum_{j=1}^{j=n} T_{ij} V_j + w_{i0} \tag{3}$$

Thus the synaptic weights are an exact copy of the patterns recorded at each grid point during learning and the neurons can be thought of as processors which compute distances to those patterns. During navigation, the nearest node is identified with a network of $k$ neurons evaluating $k$ discriminant functions in parallel, followed by a "winner-take-all" network to pick the maximum $g_i(\mathbf{x})$. This is the well-known implementation of the nearest-neighbour classifier on a neural network architecture. Since the $\mathbf{u}_i$'s are *analog* input vectors, then the synaptic weights $T_{ij}$ will also be analog quantities and this leads to a very efficient use of the pulse-stream analog VLSI technology which we have recently developed for the implementation of neural networks (Murray *et al*, 1990).

With pulse-stream arithmetic, analog computation is performed under digital control. The neural states are represented by *pulse rates* and synaptic multiplication is achieved by pulse width modulation. This allows very compact, fully-

programmable, synapse circuits to be designed (3 or 4 transistors per synapse). We have already applied one set of our working chips to the nearest-neighbour classification task described in this Section. They were evaluated on a 24-node test environment and full results have been reported elsewhere (Brownlow, Tarassenko and Murray, 1990). It was found that the $\sum T_{ij} V_j$ scalar products evaluated by our VLSI chips on this test problem were always within 1.2% of those computed on a SUN 3/80 workstation.

# 5   OBSTACLE DETECTION/AVOIDANCE

A more appropriate name for this module may be that of *local navigation*. The module will rely on optical flow information derived from a number of *fixed* optical sensors mounted on the robot platform. Each sensor will include a pulsed light source to illuminate the scene locally and the light reflected from nearby objects will be focussed onto a pair of gratings at right angles to each other, before being detected by a photodiode array. From the time derivatives of the received signals, it is possible to compute the relative velocities of nearby objects such as moving obstacles. We plan to use previous work on structure from motion to pre-process these velocity vectors and derive from them appropriate feature vectors to be used as inputs to a low-level neural network for motor control (see Figure 3 below).

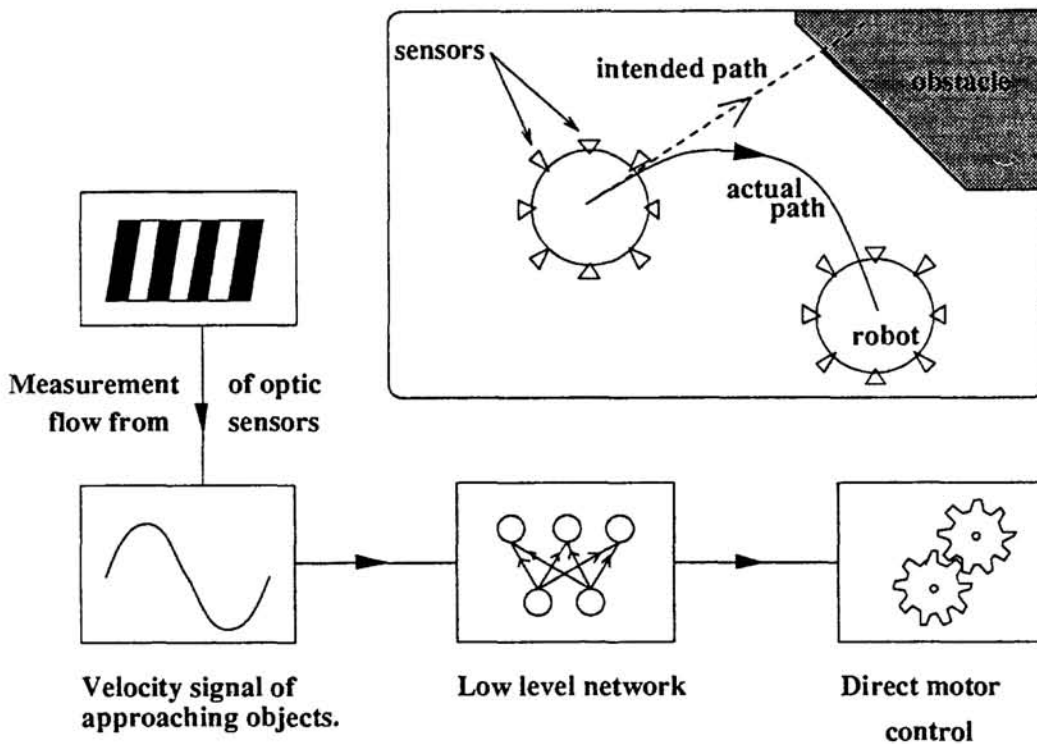

Figure 3: Sensory-motor associative network for obstacle avoidance

The obstacle avoidance network will be taught to associate appropriate motor behaviours with different types of sensory input data, for example the taking of the correct evasive action when a moving object is approaching the robot from a particular direction. This module will therefore be responsible for *path adjustment* in response to dynamic obstacles (with a bandwidth of around 100 Hz), but the path planner of Section 3 will continue to deal with *path reconfiguration* at a much lower data rate (1 Hz), once the dynamic obstacle has been avoided. Our work on this module has, so far, been mainly concerned with the design of the input sensors and associated electronics.

## 6    CONCLUSION

We have implemented the path planning and localization modules described in this paper on a SUN 4 workstation and used them to control a mobile robot platform via a radio link. This capability was demonstrated at the NIPS'90 Conference with a videotape recording of our mobile robot navigating around static obstacles in a laboratory environment, using real-time infra-red data for localization. It was possible to run the path planner in near real-time in simulation because no resistor value need be changed in a static environment; in order to achieve real-time path planning in a *dynamic* environment, however, the hardware solution of Section 3 will be mandatory. Our aim remains the implementation of all 3 modules in VLSI in order to demonstrate a fully autonomous real-time navigation system with all the sensors and hardware mounted on the robot platform.

**Acknowledgements**

We gratefully acknowledge the financial support of UK Science and Engineering Research Council and of the EEC (ESPRIT BRA). We have benefitted greatly from the help and advice of members of the Robotics Research Group, most notably Martin Adams, Gabriel Hamid and Jake Reynolds.

**References**

M.J. Brownlow, L. Tarassenko & A.F. Murray. (1990) Analogue computation using VLSI neural network devices. *Electronics Letters*, **26**(16):1297-1299.

B.K.P. Horn. (1974) Determining lightness from an image. *Computational Graphics & Image Processing*, **3**:277-299.

C.A. Mead & M.A. Mahowald. (1988) A silicon model of early visual processing. *Neural Networks*, **1**(1):91-97.

A.F. Murray, M.J. Brownlow, L. Tarassenko, A. Hamilton, I.S. Han & H.M. Reekie. (1990) Pulse-Firing Neural Chips for Hundreds of Neurons. In D.S. Touretzky (ed.), *Advances in Neural Information Processing Systems 2*, 785-792. San Mateo, CA: Morgan Kaufmann.

L. Tarassenko & A. Blake. (1991). Analogue computation of collision-free paths. To be published in: *Proceedings of 1991 IEEE Int. Conf. on Robotics & Automation*, Sacramento, CA:

## Footnotes

*Also: RSRE, Great Malvern, Worcester, WR14 3PS
